# Generalization and scaling in reinforcement learning

**David H. Ackley**
**Michael L. Littman**
Cognitive Science Research Group
Bellcore
Morristown, NJ 07960

## ABSTRACT

In *associative reinforcement learning*, an environment generates input
vectors, a learning system generates possible output vectors, and a re-
inforcement function computes feedback signals from the input-output
pairs. The task is to discover and remember input-output pairs that
generate rewards. Especially difficult cases occur when rewards are
rare, since the expected time for any algorithm can grow exponentially
with the size of the problem. Nonetheless, if a reinforcement function
possesses regularities, and a learning algorithm exploits them, learning
time can be reduced below that of non-generalizing algorithms. This
paper describes a neural network algorithm called *complementary re-
inforcement back-propagation* (*CRBP*), and reports simulation results
on problems designed to offer differing opportunities for generalization.

## 1    REINFORCEMENT LEARNING REQUIRES SEARCH

Reinforcement learning (Sutton, 1984; Barto & Anandan, 1985; Ackley, 1988; Allen,
1989) requires more from a learner than does the more familiar supervised learning
paradigm. Supervised learning supplies the correct answers to the learner, whereas
reinforcement learning requires the learner to *discover* the correct outputs before
they can be stored. The reinforcement paradigm divides neatly into search and
learning aspects: When rewarded the system makes internal adjustments to learn
the discovered input-output pair; when punished the system makes internal adjust-
ments to search elsewhere.

## 1.1  MAKING REINFORCEMENT INTO ERROR

Following work by Anderson (1986) and Williams (1988), we extend the backpropagation algorithm to associative reinforcement learning. Start with a "garden variety" backpropagation network: A vector $i$ of $n$ binary input units propagates through zero or more layers of hidden units, ultimately reaching a vector $s$ of $m$ sigmoid units, each taking continuous values in the range (0,1). Interpret each $s_j$ as the *probability* that an associated random bit $o_j$ takes on value 1. Let us call the continuous, deterministic vector $s$ the *search vector* to distinguish it from the stochastic binary output vector $o$.

Given an input vector, we forward propagate to produce a search vector $s$, and then perform $m$ independent Bernoulli trials to produce an output vector $o$. The $i - o$ pair is evaluated by the reinforcement function and reward or punishment ensues. Suppose reward occurs. We therefore want to make $o$ more likely given $i$. Backpropagation will do just that if we take $o$ as the desired target to produce an error vector $(o - s)$ and adjust weights normally.

Now suppose punishment occurs, indicating $o$ does not correspond with $i$. By choice of error vector, backpropagation allows us to push the search vector in any direction; which way should we go? In absence of problem-specific information, we cannot pick an appropriate direction with certainty. Any decision will involve assumptions. A very minimal "don't be like $o$" assumption—employed in Anderson (1986), Williams (1988), and Ackley (1989)—pushes $s$ directly *away from* $o$ by taking $(s - o)$ as the error vector. A slightly stronger "be like not-$o$" assumption—employed in Barto & Anandan (1985) and Ackley (1987)—pushes $s$ directly *toward the complement of* $o$ by taking $((1 - o) - s)$ as the error vector. Although the two approaches always agree on the signs of the error terms, they differ in magnitudes. In this work, we explore the second possibility, embodied in an algorithm called *complementary reinforcement back-propagation (CRBP)*.

Figure 1 summarizes the *CRBP* algorithm. The algorithm in the figure reflects three modifications to the basic approach just sketched. First, in step 2, instead of using the $s_j$'s directly as probabilities, we found it advantageous to "stretch" the values using a parameter $\nu$. When $\nu < 1$, it is not necessary for the $s_j$'s to reach zero or one to produce a deterministic output. Second, in step 6, we found it important to use a smaller learning rate for punishment compared to reward. Third, consider step 7: Another forward propagation is performed, another stochastic binary output vector $o^*$ is generated (using the procedure from step 2), and $o^*$ is compared to $o$. If they are identical and punishment occurred, or if they are different and reward occurred, then another error vector is generated and another weight update is performed. This loop continues until a different output is generated (in the case of failure) or until the original output is regenerated (in the case of success). This modification improved performance significantly, and added only a small percentage to the total number of weight updates performed.

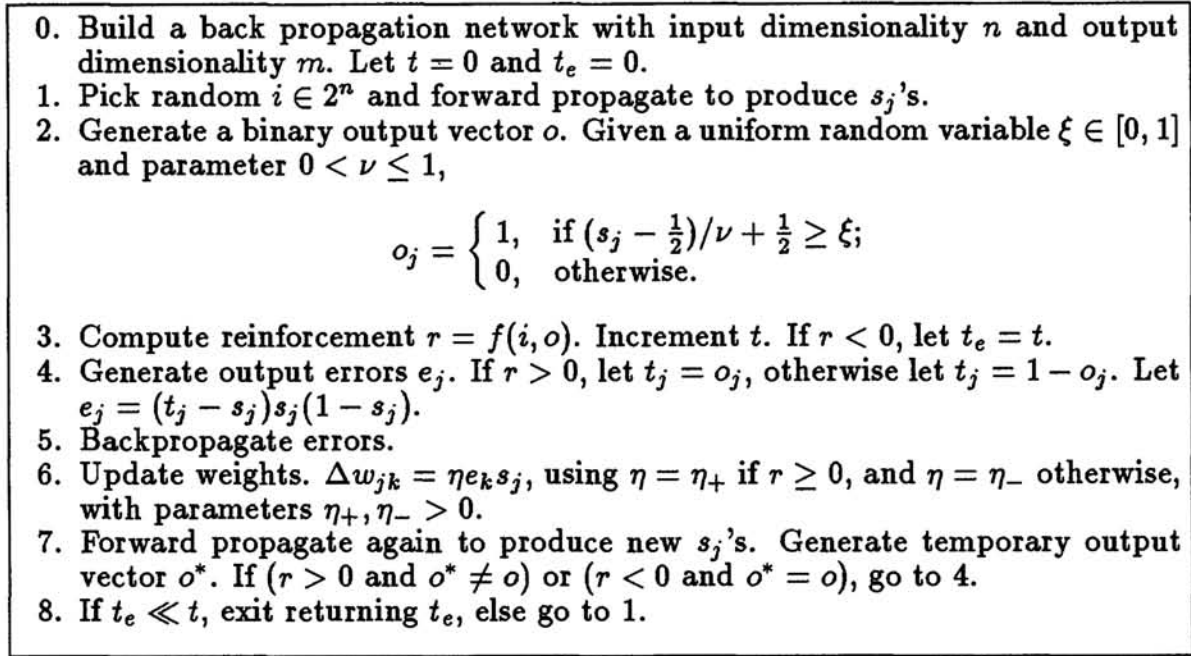

0. Build a back propagation network with input dimensionality $n$ and output dimensionality $m$. Let $t = 0$ and $t_e = 0$.
1. Pick random $i \in 2^n$ and forward propagate to produce $s_j$'s.
2. Generate a binary output vector $o$. Given a uniform random variable $\xi \in [0, 1]$ and parameter $0 < \nu \leq 1$,

$$o_j = \begin{cases} 1, & \text{if } (s_j - \frac{1}{2})/\nu + \frac{1}{2} \geq \xi; \\ 0, & \text{otherwise.} \end{cases}$$

3. Compute reinforcement $r = f(i, o)$. Increment $t$. If $r < 0$, let $t_e = t$.
4. Generate output errors $e_j$. If $r > 0$, let $t_j = o_j$, otherwise let $t_j = 1 - o_j$. Let $e_j = (t_j - s_j)s_j(1 - s_j)$.
5. Backpropagate errors.
6. Update weights. $\Delta w_{jk} = \eta e_k s_j$, using $\eta = \eta_+$ if $r \geq 0$, and $\eta = \eta_-$ otherwise, with parameters $\eta_+, \eta_- > 0$.
7. Forward propagate again to produce new $s_j$'s. Generate temporary output vector $o^*$. If $(r > 0$ and $o^* \neq o)$ or $(r < 0$ and $o^* = o)$, go to 4.
8. If $t_e \ll t$, exit returning $t_e$, else go to 1.

Figure 1: Complementary Reinforcement Back Propagation—CRBP

## 2   ON-LINE GENERALIZATION

When there are many possible outputs and correct pairings are rare, the computational cost associated with the search for the correct answers can be profound. The search for correct pairings will be accelerated if the search strategy can effectively *generalize* the reinforcement received on one input to others. The speed of an algorithm on a given problem relative to non-generalizing algorithms provides a measure of generalization that we call *on-line generalization*.

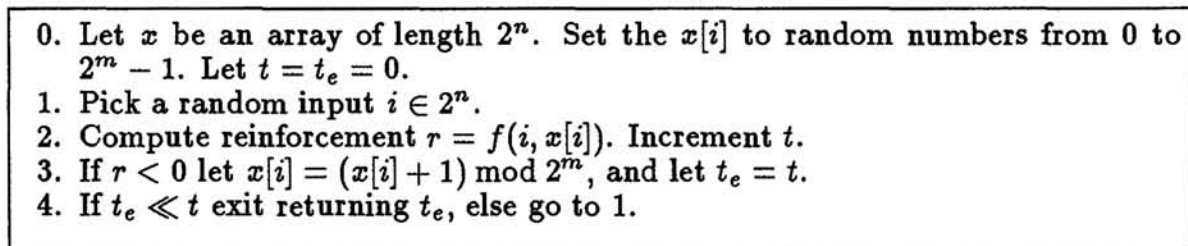

0. Let $x$ be an array of length $2^n$. Set the $x[i]$ to random numbers from 0 to $2^m - 1$. Let $t = t_e = 0$.
1. Pick a random input $i \in 2^n$.
2. Compute reinforcement $r = f(i, x[i])$. Increment $t$.
3. If $r < 0$ let $x[i] = (x[i] + 1) \bmod 2^m$, and let $t_e = t$.
4. If $t_e \ll t$ exit returning $t_e$, else go to 1.

Figure 2: The Table Lookup Reference Algorithm $T_{\text{ref}}(f, n, m)$

Consider the table-lookup algorithm $T_{\text{ref}}(f, n, m)$ summarized in Figure 2. In this algorithm, a separate storage location is used for each possible input. This prevents the memorization of one $i - o$ pair from interfering with any other. Similarly, the selection of a candidate output vector depends only on the slot of the table corresponding to the given input. The learning speed of $T_{\text{ref}}$ depends only on the input and output dimensionalities and the number of correct outputs associated with each input. When a problem possesses $n$ input bits and $n$ output bits, and
there is only one correct output vector for each input vector, $T_{\text{ref}}$ runs in about $4^n$
time (counting each input-output judgment as one.) In such cases one expects to
take at least $2^{n-1}$ just to find *one* correct $i - o$ pair, so exponential time cannot be
avoided without *a priori* information. How does a generalizing algorithm such as
*CRBP* compare to $T_{\text{ref}}$?

## 3   SIMULATIONS ON SCALABLE PROBLEMS

We have tested *CRBP* on several simple problems designed to offer varying degrees
and types of generalization. In all of the simulations in this section, the following
details apply: Input and output bit counts are equal ($n$). Parameters are dependent
on $n$ but independent of the reinforcement function $f$. $\eta_+$ is hand-picked for each
$n$,[1] $\eta_- = \eta_+/10$ and $\nu = 0.5$. All data points are medians of five runs. The stopping
criterion $t_e \ll t$ is interpreted as $t_e + \max(2000, 2^{n+1}) < t$. The fit lines in the figures
are least squares solutions to $a \times b^n$, to two significant digits.

As a notational convenience, let $c = \frac{1}{n} \sum_{j=1}^{n} i_j$ — the fraction of ones in the input.

### 3.1   $n$-MAJORITY

Consider this "majority rules" problem: [if $c > \frac{1}{2}$ then $o = 1^n$ else $o = 0^n$]. The $i-o$
mapping is many-to-1. This problem provides an opportunity for what Anderson
(1986) called "output generalization": since there are only two correct output states,
every pair of output bits are completely correlated in the cases when reward occurs.

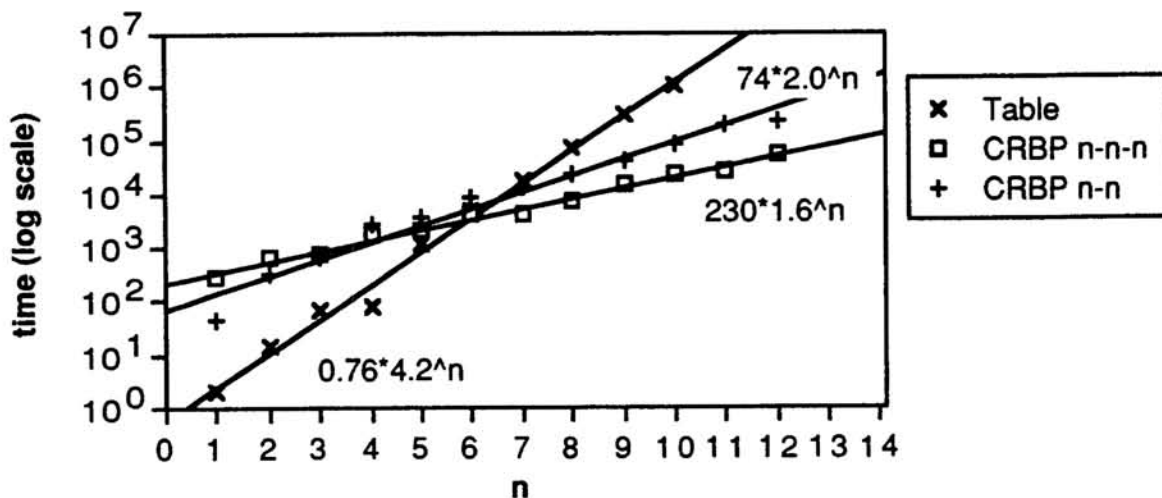

**Figure 3:** The $n$-majority problem

Figure 3 displays the simulation results. Note that although $T_{\text{ref}}$ is faster than
*CRBP* at small values of $n$, *CRBP*'s slower growth rate ($1.6^n$ vs $4.2^n$) allows it to
cross over and begin outperforming $T_{\text{ref}}$ at about 6 bits. Note also—in violation of

[1]For $n = 1$ to 12, we used $\eta_+ = \{2.000, 1.550, 1.130, 0.979, 0.783, 0.709, 0.623, 0.525, 0.280,$
$0.219, 0.170, 0.121\}$.

some conventional wisdom—that although $n$-majority is a linearly separable problem, the performance of *CRBP* with hidden units is better than without. Hidden units can be helpful—even on linearly separable problems—when there are opportunities for output generalization.

## 3.2  $n$-COPY AND THE $2^k$-ATTRACTORS FAMILY

As a second example, consider the $n$-copy problem: $[o = i]$. The $i-o$ mapping is now 1-1, and the values of output bits in rewarding states are completely uncorrelated, but the value of each output bit is completely correlated with the value of the corresponding input bit. Figure 4 displays the simulation results. Once again, at

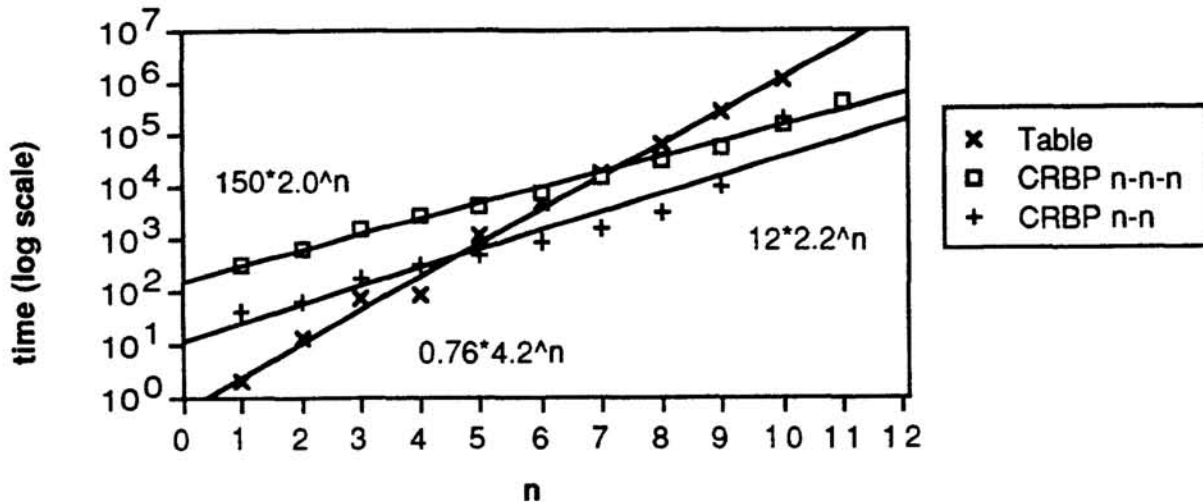

**Figure 4:** The $n$-copy problem

low values of $n$, $T_{\text{ref}}$ is faster, but *CRBP* rapidly overtakes $T_{\text{ref}}$ as $n$ increases. In $n$-copy, unlike $n$-majority, *CRBP* performs better without hidden units.

The $n$-majority and $n$-copy problems are extreme cases of a spectrum. $n$-majority can be viewed as a "2-attractors" problem in that there are only two correct outputs—all zeros and all ones—and the correct output is the one that $i$ is closer to in hamming distance. By dividing the input and output bits into two groups and performing the majority function independently on each group, one generates a "4-attractors" problem. In general, by dividing the input and output bits into $1 \le k \le n$ groups, one generates a "$2^k$-attractors" problem. When $k = 1$, $n$-majority results, and when $k = n$, $n$-copy results.

Figure 5 displays simulation results on the $n = 8$-bit problems generated when $k$ is varied from 1 to $n$. The advantage of hidden units for low values of $k$ is evident, as is the advantage of "shortcut connections" (direct input-to-output weights) for larger values of $k$. Note also that combination of both hidden units and shortcut connections performs better than either alone.

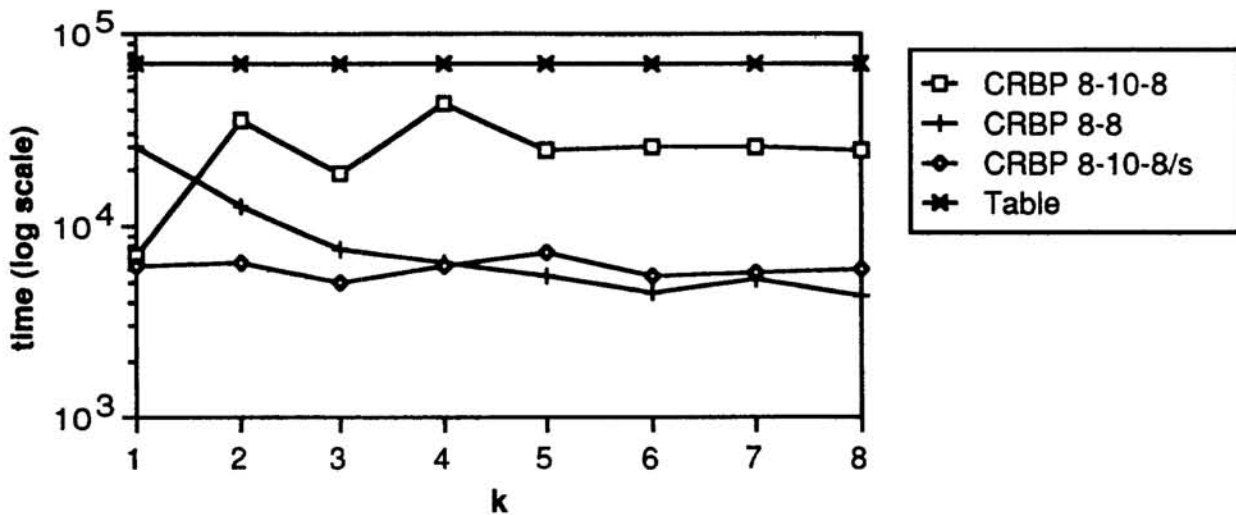

Figure 5: The $2^k$-attractors family at $n = 8$

## 3.3  $n$-EXCLUDED MIDDLE

All of the functions considered so far have been linearly separable. Consider this "folded majority" function: [if $\frac{1}{3} < c \le \frac{2}{3}$ then $o = 0^n$ else $o = 1^n$]. Now, like $n$-majority, there are only two rewarding output states, but the determination of which output state is correct is not linearly separable in the input space. When $n = 2$, the $n$-excluded middle problem yields the EQV (i.e., the complement of XOR) function, but whereas functions such as $n$-parity [if $nc$ is even then $o = 0^n$ else $o = 1^n$] get more non-linear with increasing $n$, $n$-excluded middle does not.

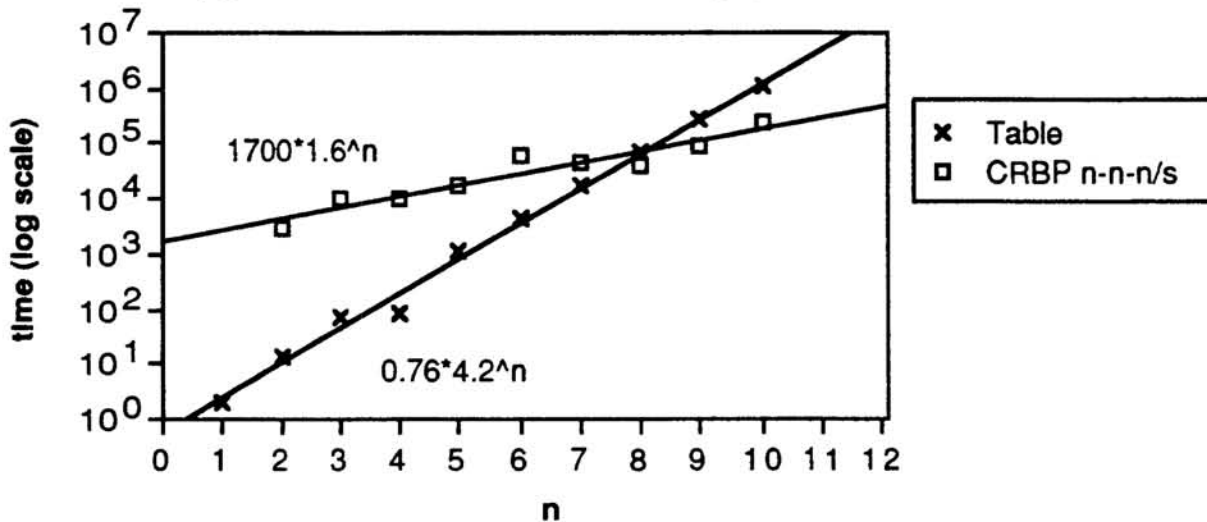

Figure 6: The $n$-excluded middle problem

Figure 6 displays the simulation results. *CRBP* is slowed somewhat compared to the linearly separable problems, yielding a higher "cross over point" of about 8 bits.

## 4  STRUCTURING DEGENERATE OUTPUT SPACES

All of the scaling problems in the previous section are designed so that there is a single correct output for each possible input. This allows for difficult problems even at small sizes, but it rules out an important aspect of generalizing algorithms for associative reinforcement learning: If there are multiple satisfactory outputs for given inputs, a generalizing algorithm may *impose structure* on the mapping it produces.

We have two demonstrations of this effect, "Bit Count" and "Inverse Arithmetic." The Bit Count problem simply states that the number of 1-bits in the output should equal the number of 1-bits in the input. When $n = 9$, $T_{\text{ref}}$ rapidly finds solutions involving hundreds of different output patterns. $CRBP$ is slower—especially with relatively few hidden units—but it regularly finds solutions involving just 10 output patterns that form a sequence from $0^9$ to $1^9$ with one bit changing per step.

$$
\begin{array}{llll}
0 + 0 \times 4 = 0 & 0 + 2 \times 4 = 8 & 0 + 4 \times 4 = 16 & 0 + 6 \times 4 = 24 \\
1 + 0 \times 4 = 1 & 1 + 2 \times 4 = 9 & 1 + 4 \times 4 = 17 & 1 + 6 \times 4 = 25 \\
2 + 0 \times 4 = 2 & 2 + 2 \times 4 = 10 & 2 + 4 \times 4 = 18 & 2 + 6 \times 4 = 26 \\
3 + 0 \times 4 = 3 & 3 + 2 \times 4 = 11 & 3 + 4 \times 4 = 19 & 3 + 6 \times 4 = 27 \\
\\
4 + 0 \times 4 = 4 & 4 + 2 \times 4 = 12 & 4 + 4 \times 4 = 20 & 4 + 6 \times 4 = 28 \\
5 + 0 \times 4 = 5 & 5 + 2 \times 4 = 13 & 5 + 4 \times 4 = 21 & 5 + 6 \times 4 = 29 \\
6 + 0 \times 4 = 6 & 6 + 2 \times 4 = 14 & 6 + 4 \times 4 = 22 & 6 + 6 \times 4 = 30 \\
7 + 0 \times 4 = 7 & 7 + 2 \times 4 = 15 & 7 + 4 \times 4 = 23 & 7 + 6 \times 4 = 31 \\
\\
2 + 2 - 4 = 0 & 2 + 2 + 4 = 8 & 6 + 6 + 4 = 16 & 0 + 6 \times 4 = 24 \\
3 + 2 - 4 = 1 & 3 + 2 + 4 = 9 & 7 + 6 + 4 = 17 & 1 + 6 \times 4 = 25 \\
2 + 2 \div 4 = 2 & 2 + 2 \times 4 = 10 & 2 + 4 \times 4 = 18 & 2 + 6 \times 4 = 26 \\
3 + 2 \div 4 = 3 & 3 + 2 \times 4 = 11 & 3 + 4 \times 4 = 19 & 3 + 6 \times 4 = 27 \\
\\
6 + 2 - 4 = 4 & 6 + 2 + 4 = 12 & 4 \times 4 + 4 = 20 & 4 + 6 \times 4 = 28 \\
7 + 2 - 4 = 5 & 7 + 2 + 4 = 13 & 5 + 4 \times 4 = 21 & 5 + 6 \times 4 = 29 \\
6 + 2 \div 4 = 6 & 6 + 2 \times 4 = 14 & 6 + 4 \times 4 = 22 & 6 + 6 \times 4 = 30 \\
7 + 2 \div 4 = 7 & 7 + 2 \times 4 = 15 & 7 + 4 \times 4 = 23 & 7 + 6 \times 4 = 31 \\
\end{array}
$$

**Figure 7:** Sample $CRBP$ solutions to Inverse Arithmetic

The Inverse Arithmetic problem can be summarized as follows: Given $i \in 2^5$, find $x, y, z \in 2^3$ and $\circ, \diamond \in \{+_{(00)}, -_{(01)}, \times_{(10)}, \div_{(11)}\}$ such that $x \circ y \diamond z = i$. In all there are 13 bits of output, interpreted as three 3-bit binary numbers and two 2-bit operators, and the task is to pick an output that evaluates to the given 5-bit binary input under the usual rules: operator precedence, left-right evaluation, integer division, and division by zero fails.

As shown in Figure 7, $CRBP$ sometimes solves this problem essentially by discovering positional notation, and sometimes produces less-globally structured solutions, particularly as outputs for lower-valued $i$'s, which have a wider range of solutions.

# 5  CONCLUSIONS

Some basic concepts of supervised learning appear in different guises when the paradigm of reinforcement learning is applied to large output spaces. Rather than a "learning phase" followed by a "generalization test," in reinforcement learning the *search problem is a generalization test*, performed simultaneously with learning. Information is put to work as soon as it is acquired.

The problem of of "overfitting" or "learning the noise" seems to be less of an issue, since learning stops automatically when consistent success is reached. In experiments not reported here we gradually increased the number of hidden units on the 8-bit copy problem from 8 to 25 without observing the performance decline associated with "too many free parameters."

The $2^k$-attractors (and $2^k$-folds—generalizing Excluded Middle) families provide a starter set of sample problems with easily understood and distinctly different extreme cases.

In degenerate output spaces, generalization decisions can be seen directly in the discovered mapping. Network analysis is not required to "see how the net does it."

The possibility of ultimately generating useful new knowledge via reinforcement learning algorithms cannot be ruled out.

**References**

Ackley, D.H. (1987) *A connectionist machine for genetic hillclimbing.* Boston, MA: Kluwer Academic Press.

Ackley, D.H. (1989) Associative learning via inhibitory search. In D.S. Touretzky (ed.), *Advances in Neural Information Processing Systems 1*, 20–28. San Mateo, CA: Morgan Kaufmann.

Allen, R.B. (1989) Developing agent models with a neural reinforcement technique. *IEEE Systems, Man, and Cybernetics Conference.* Cambridge, MA.

Anderson, C.W. (1986) Learning and problem solving with multilayer connectionist systems. University of Mass. Ph.D. dissertation. COINS TR 86–50. Amherst, MA.

Barto, A.G. (1985) Learning by statistical cooperation of self-interested neuron-like computing elements. *Human Neurobiology*, 4:229–256.

Barto, A.G., & Anandan, P. (1985) Pattern recognizing stochastic learning automata. *IEEE Transactions on Systems, Man, and Cybernetics, 15*, 360–374.

Rumelhart, D.E., Hinton, G.E., & Williams, R.J. (1986) Learning representations by back-propagating errors. *Nature, 323*, 533–536.

Sutton, R.S. (1984) Temporal credit assignment in reinforcement learning. University of Mass. Ph.D. dissertation. COINS TR 84–2. Amherst, MA.

Williams, R.J. (1988) Toward a theory of reinforcement–learning connectionist systems. College of Computer Science of Northeastern University Technical Report NU–CCS–88–3. Boston, MA.
